# Probing the Compositionality of Intuitive Functions

**Eric Schulz**
University College London
e.schulz@cs.ucl.ac.uk

**Joshua B. Tenenbaum**
MIT
jbt@mit.edu

**David Duvenaud**
University of Toronto
duvenaud@cs.toronto.edu

**Maarten Speekenbrink**
University College London
m.speekenbrink@ucl.ac.uk

**Samuel J. Gershman**
Harvard University
gershman@fas.harvard.edu

## Abstract

How do people learn about complex functional structure? Taking inspiration from other areas of cognitive science, we propose that this is accomplished by harnessing compositionality: complex structure is decomposed into simpler building blocks. We formalize this idea within the framework of Bayesian regression using a grammar over Gaussian process kernels. We show that participants prefer compositional over non-compositional function extrapolations, that samples from the human prior over functions are best described by a compositional model, and that people perceive compositional functions as more predictable than their non-compositional but otherwise similar counterparts. We argue that the compositional nature of intuitive functions is consistent with broad principles of human cognition.

## 1 Introduction

Function learning underlies many intuitive judgments, such as the perception of time, space and number. All of these tasks require the construction of mental representations that map inputs to outputs. Since the space of such mappings is infinite, inductive biases are necessary to constrain the plausible inferences. What is the nature of human inductive biases over functions?

It has been suggested that Gaussian processes (GPs) provide a good characterization of these inductive biases [15]. As we describe more formally below, GPs are distributions over functions that can encode properties such as smoothness, linearity, periodicity, and other inductive biases indicated by research on human function learning [5, 3]. Lucas et al. [15] showed how Bayesian inference with GP priors can unify previous rule-based and exemplar-based theories of function learning [18].

A major unresolved question is how people deal with complex functions that are not easily captured by any simple GP. Insight into this question is provided by the observation that many complex functions encountered in the real world can be broken down into compositions of simpler functions [6, 11]. We pursue this idea theoretically and experimentally, by first defining a hypothetical compositional grammar for intuitive functions (based on [6]) and then investigating whether this grammar quantitatively predicts human function learning performance. We compare the compositional model to a flexible non-compositional model (the spectral mixture representation proposed by [21]). Both models use Bayesian inference to reason about functions, but differ in their inductive biases.

We show that (a) participants prefer compositional pattern extrapolations in both forced choice and manual drawing tasks; (b) samples elicited from participants' priors over functions are more consistent with the compositional grammar; and (c) participants perceive compositional functions as more predictable than non-compositional ones. Taken together, these findings provide support for the compositional nature of intuitive functions.

## 2 Gaussian process regression as a theory of intuitive function learning

A GP is a collection of random variables, any finite subset of which are jointly Gaussian-distributed (see [18] for an introduction). A GP can be expressed as a distribution over functions: $f \sim \mathcal{GP}(m, k)$, where $m(\mathbf{x}) = \mathbb{E}[f(\mathbf{x})]$ is a mean function modeling the expected output of the function given input $\mathbf{x}$, and $k(\mathbf{x}, \mathbf{x}') = \mathbb{E}[(f(\mathbf{x}) - m(\mathbf{x}))(f(\mathbf{x}') - m(\mathbf{x}'))]$ is a kernel function modeling the covariance between points. Intuitively, the kernel encodes an inductive bias about the expected smoothness of functions drawn from the GP. To simplify exposition, we follow standard convention in assuming a constant mean of 0.

Conditional on data $\mathcal{D} = \{\mathbf{X}, \mathbf{y}\}$, where $y_n \sim \mathcal{N}(f(\mathbf{x}_n), \sigma^2)$, the posterior predictive distribution for a new input $\mathbf{x}_*$ is Gaussian with mean and variance given by:

$$\mathbb{E}[f(\mathbf{x}_\star)|\mathcal{D}] = \mathbf{k}_\star^\top (\mathbf{K} + \sigma^2 \mathbf{I})^{-1} \mathbf{y} \tag{1}$$

$$\mathbb{V}[f(\mathbf{x}_\star)|\mathcal{D}] = k(\mathbf{x}_\star, \mathbf{x}_\star) - \mathbf{k}_\star^\top (\mathbf{K} + \sigma^2 \mathbf{I})^{-1} \mathbf{k}_\star, \tag{2}$$

where $\mathbf{K}$ is the $N \times N$ matrix of covariances evaluated at each input in $\mathbf{X}$ and $\mathbf{k}_\star = [k(\mathbf{x}_1, \mathbf{x}_*), \ldots, k(\mathbf{x}_N, \mathbf{x}_*)]$.

As pointed out by Griffiths et al. [10] (see also [15]), the predictive distribution can be viewed as an exemplar (similarity-based) model of function learning [5, 16], since it can be written as a linear combination of the covariance between past and current inputs:

$$f(\mathbf{x}_*) = \sum_{n=1}^{N} \alpha_n k(\mathbf{x}_n, \mathbf{x}_\star) \tag{3}$$

with $\alpha = (\mathbf{K} + \sigma^2 \mathbf{I})^{-1} \mathbf{y}$. Equivalently, by Mercer's theorem any positive definite kernel can be expressed as an outer product of feature vectors:

$$k(\mathbf{x}, \mathbf{x}') = \sum_{d=1}^{\infty} \lambda_d \phi_d(\mathbf{x}) \phi_d(\mathbf{x}'), \tag{4}$$

where $\{\phi_d(\mathbf{x})\}$ are the eigenfunctions of the kernel and $\{\lambda_d\}$ are the eigenvalues. The posterior predictive mean is a linear combination of the features, which from a psychological perspective can be thought of as encoding "rules" mapping inputs to outputs [4, 14]. Thus, a GP can be expressed as both an exemplar (similarity-based) model and a feature (rule-based) model, unifying the two dominant classes of function learning theories in cognitive science [15].

## 3 Structure learning with Gaussian processes

So far we have assumed a fixed kernel function. However, humans can adapt to a wide variety of structural forms [13, 8], suggesting that they have the flexibility to learn the kernel function from experience. The key question addressed in this paper is what space of kernels humans are optimizing over—how rich is their representational vocabulary? This vocabulary will in turn act as an inductive bias, making some functions easier to learn, and other functions harder to learn.

Broadly speaking, there are two approaches to parameterizing the kernel space: a fixed functional form with continuous parameters, or a combinatorial space of functional forms. These approaches are not mutually exclusive; indeed, the success of the combinatorial approach depends on optimizing the continuous parameters for each form. Nonetheless, this distinction is useful because it allows us to separate different forms of functional complexity. A function might have internal structure such that when this structure is revealed, the apparent functional complexity is significantly reduced. For example, a function composed of many piecewise linear segments might have a long description length under a typical continuous parametrization (e.g., the radial basis kernel described below), because it violates the smoothness assumptions of the prior. However, conditional on the change-points between segments, the function can be decomposed into independent parts each of which is well-described by a simple continuous parametrization. If internally structured functions are "natural kinds," then the combinatorial approach may be a good model of human intuitive functions.

In the rest of this section, we describe three kernel parameterizations. The first two are continuous, differing in their expressiveness. The third one is combinatorial, allowing it to capture complex patterns by composing simpler kernels. For all kernels, we take the standard approach of choosing the parameter values that optimize the log marginal likelihood.

### 3.1 Radial basis kernel

The radial basis kernel is a commonly used kernel in machine learning applications, embodying the assumption that the covariance between function values decays exponentially with input distance:

$$k(\boldsymbol{x}, \boldsymbol{x}') = \theta^2 \exp\left(-\frac{|\boldsymbol{x} - \boldsymbol{x}'|^2}{2l^2}\right), \tag{5}$$

where $\theta$ is a scaling parameter and $l$ is a length-scale parameter. This kernel assumes that the same smoothness properties apply globally for all inputs. It provides a standard baseline to compare with more expressive kernels.

### 3.2 Spectral mixture kernel

The second approach is based on the fact that any stationary kernel can be expressed as an integral using Bochner's theorem. Letting $\boldsymbol{\tau} = |\boldsymbol{x} - \boldsymbol{x}'| \in \mathbb{R}^P$, then

$$k(\boldsymbol{\tau}) = \int_{\mathbb{R}^P} e^{2\pi i \boldsymbol{s}^\top \boldsymbol{\tau}} \psi(\mathrm{d}\boldsymbol{s}). \tag{6}$$

If $\psi$ has a density $S(\boldsymbol{s})$, then $S$ is the spectral density of $k$; $S$ and $k$ are thus Fourier duals [18]. This means that a spectral density fully defines the kernel and that furthermore every stationary kernel can be expressed as a spectral density. Wilson & Adams [21] showed that the spectral density can be approximated by a mixture of $Q$ Gaussians, such that

$$k(\boldsymbol{\tau}) = \sum_{q=1}^{Q} w_q \prod_{p=1}^{P} \exp\left(-2\pi^2 \tau_p^2 \upsilon_q^p\right) \cos\left(2\pi \tau_p \mu_q^{(p)}\right) \tag{7}$$

Here, the $q$th component has mean vector $\mu_q = \left(\mu_q^{(1)}, \ldots, \mu_q^{(P)}\right)$ and a covariance matrix $\mathbf{M}_q = \mathrm{diag}\left(\upsilon_q^{(1)}, \ldots, \upsilon_q^{(P)}\right)$. The result is a non-parametric approach to Gaussian process regression, in which complex kernels are approximated by mixtures of simpler ones. This approach is appealing when simpler kernels fail to capture functional structure. Its main drawback is that because structure is captured implicitly via the spectral density, the building blocks are psychologically less intuitive: humans appear to have preferences for linear [12] and periodic [1] functions, which are not straightforwardly encoded in the spectral mixture (though of course the mixture can approximate these functions). Since the spectral kernel has been successfully applied to reverse engineer human kernels [22], it is a useful reference of comparison to more structured compositional approaches.

### 3.3 Compositional kernel

As positive semidefinite kernels are closed under addition and multiplication, we can create richly structured and interpretable kernels from well understood base components. For example, by summing kernels, we can model the data as a superposition of independent functions. Figure 1 shows an example of how different kernels (radial basis, linear, periodic) can be combined. Table 1 summarizes the kernels used in our grammar.

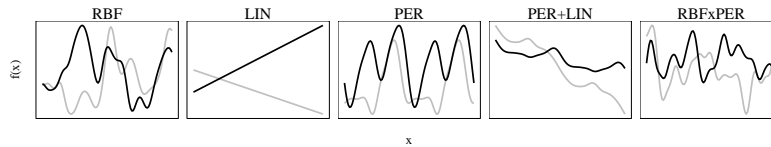

Figure 1: Examples of base and compositional kernels.

Many other compositional grammars are possible. For example, we could have included a more diverse set of kernels, and other composition operators (e.g., convolution, scaling) that generate valid kernels. However, we believe that our simple grammar is a useful starting point, since the components are intuitive and likely to be psychologically plausible. For tractability, we fix the maximum number of combined kernels to 3. Additionally, we do not allow for repetition of kernels in order to restrict the complexity of the kernel space.

| Linear | Radial basis function | Periodic |
|---|---|---|
| $k(\boldsymbol{x}, \boldsymbol{x}') = (\boldsymbol{x} - \theta_1)(\boldsymbol{x}' - \theta_1)$ | $k(\boldsymbol{\tau}) = \theta_2^2 \exp\left(-\frac{(\boldsymbol{\tau})^2}{2\theta_3^2}\right)$ | $k(\boldsymbol{\tau}) = \theta_4^2 \exp\left(-\frac{2\sin^2(\pi\boldsymbol{\tau}\theta_5)}{\theta_6^2}\right)$ |

Table 1: Utilized base kernels in our compositional grammar. $\boldsymbol{\tau} = |\boldsymbol{x} - \boldsymbol{x}'|$

.

# 4    Experiment 1: Extrapolation

The first experiment assessed whether people prefer compositional over non-compositional extrapolations. In experiment 1a, functions were sampled from a compositional GP and different extrapolations (mean predictions) were produced using each of the aforementioned kernels. Participants were then asked to choose among the 3 different extrapolations for a given function (see Figure 2). In detail, the outputs for $x_{\text{learn}} = [0, 0.1, \cdots, 7]$ were used as a training set to which all three kernels were fitted and then used to generate predictions for the test set $x_{\text{test}} = [7.1, 7.2, \cdots, 10]$. Their mean predictions were then used to generate one plot for every approach that showed the learned input as a blue line and the extrapolation as a red line. The procedure was repeated for 20 different compositional functions.

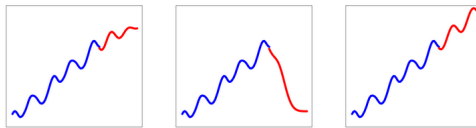

Figure 2: Screen shot of first choice experiment. Predictions in this example (from left to right) were generated by a spectral mixture, a radial basis, and a compositional kernel.

52 participants (mean age=36.15, SD = 9.11) were recruited via Amazon Mechanical Turk and received \$0.5 for their participation. Participants were asked to select one of 3 extrapolations (displayed as red lines) they thought best completed a given blue line. Results showed that participants chose compositional predictions 69%, spectral mixture predictions 17%, and radial basis predictions 14% of the time. Overall, the compositional predictions were chosen significantly more often than the other two ($\chi^2 = 591.2$, $p < 0.01$) as shown in Figure 3a.

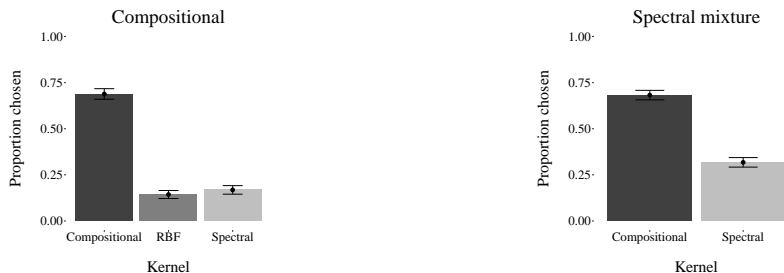

(a) Choice proportion for compositional ground truth. (b) Choice proportion for spectral mixture ground truth.

Figure 3: Results of extrapolation experiments. Error bars represent the standard error of the mean.

In experiment 1b, again 20 functions were sampled but this time from a spectral mixture kernel and 65 participants (mean age=30, SD = 9.84) were asked to choose among either compositional or spectral mixture extrapolations and received \$0.5 as before. Results (displayed in Figure 3b) showed that participants again chose compositional extrapolations more frequently (68% vs. 32%, $\chi^2 = 172.8$, $p < 0.01$), even if the ground truth happened to be generated by a spectral mixture kernel. Thus, people seem to prefer compositional over non-compositional extrapolations in forced choice extrapolation tasks.

# 5 Markov chain Monte Carlo with people

In a second set of experiments, we assessed participants' inductive biases directly using a Markov chain Monte Carlo with People (MCMCP) approach [19]. Participants accept or reject proposed extrapolations, effectively simulating a Markov chain whose stationary distribution is in this case the posterior predictive. Extrapolations from all possible kernel combinations (up to 3 combined kernels) were generated and stored a priori. These were then used to generate plots of different proposal extrapolations (as in the previous experiment). On each trial, participants chose between their most recently accepted extrapolation and a new proposal.

## 5.1 Experiment 2a: Compositional ground truth

In the first MCMCP experiment, we sampled functions from compositional kernels. Eight different functions were sampled from various compositional kernels, the input space was split into training and test sets, and then all kernel combinations were used to generate extrapolations. Proposals were sampled uniformly from this set. 51 participants with an average age of 32.55 (SD = 8.21) were recruited via Amazon's Mechanical Turk and paid $1. There were 8 blocks of 30 trials, where each block corresponded to a single training set. We calculated the average proportion of accepted kernels over the last 5 trials, as shown in Figure 4.

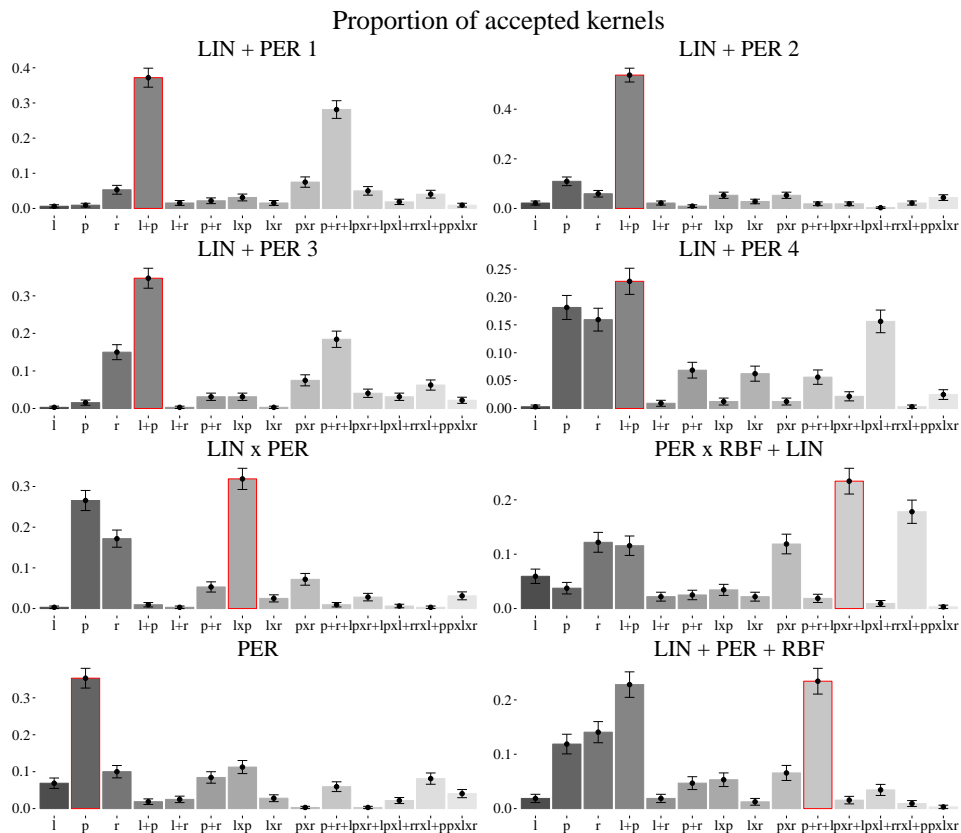

Figure 4: Proportions of chosen predictions over last 5 trials. Generating kernel marked in red.

In all cases participants' subjective probability distribution over kernels corresponded well with the data-generating kernels. Moreover, the inverse marginal likelihood, standardized over all kernels, correlated highly with the subjective beliefs assessed by MCMCP ($\rho = 0.91$, $p < .01$). Thus, participants seemed to converge to sensible structures when the functions were generated by compositional kernels.

## 5.2 Experiment 2b: Naturalistic functions

The second MCMCP experiment assessed what structures people converged to when faced with real world data. 51 participants with an average age of 32.55 (SD = 12.14) were recruited via Amazon Mechanical Turk and received $1 for their participation. The functions were an airline passenger data set, volcano $CO_2$ emission data, the number of gym memberships over 5 years, and the number of times people googled the band "Wham!" over the last 8 years; all shown in Figure 5a. Participants were not told any information about the data set (including input and output descriptions) beyond the input-output pairs. As periodicity in the real world is rarely ever purely periodic, we adapted the periodic component of the grammar by multiplying a periodic kernel with a radial basis kernel, thereby locally smoothing the periodic part of the function.[1] Apart from the different training sets, the procedure was identical to the last experiment.

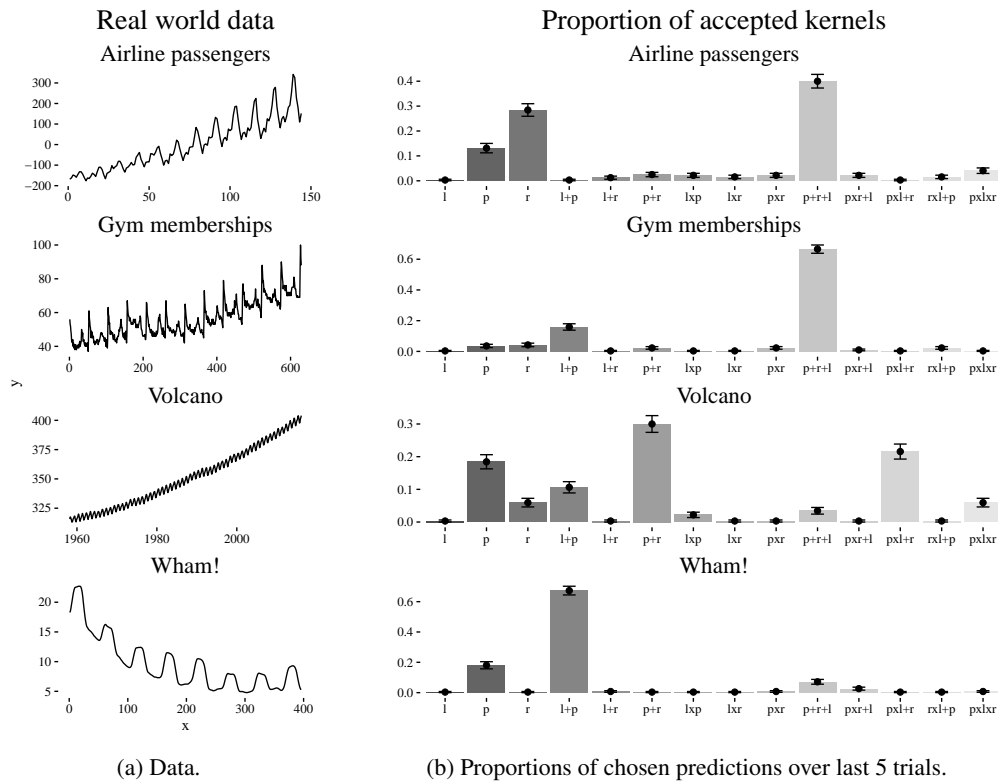

(a) Data.  (b) Proportions of chosen predictions over last 5 trials.

Figure 5: Real world data and MCMCP results. Error bars represent the standard error of the mean.

Results are shown in Figure 5b, demonstrating that participants converged to intuitively plausible patterns. In particular, for both the volcano and the airline passenger data, participants converged to compositions resembling those found in previous analyses [6]. The correlation between the mean proportion of accepted predictions and the inverse standardized marginal likelihoods of the different kernels was again significantly positive ($\rho = 0.83$, $p < .01$).

## 6 Experiment 3: Manual function completion

In the next experiment, we let participants draw the functions underlying observed data manually. As all of the prior experiments asked participants to judge between "pre-generated" predictions of functions, we wanted to compare this to how participants generate predictions themselves. On each round of the experiment, functions were sampled from the compositional grammar, the number of points to be presented on each trial was sampled uniformly between 100 and 200, and the noise variance was sampled uniformly between 0 and 25. Finally, the size of an unobserved region of the

function was sampled to lie between 5 and 50. Participants were asked to manually draw the function best describing observed data and to inter- and extrapolate this function in two unobserved regions. A screen shot of the experiment is shown in Figure 6.

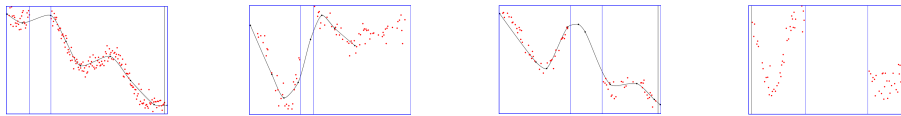

Figure 6: Manual pattern completion experiment. Extrapolation region is delimited by vertical lines.

36 participants with a mean age of 30.5 (SD = 7.15) were recruited from Amazon Mechanical Turk and received \$2 for their participation. Participants were asked to draw lines in a cloud of dots that they thought best described the given data. To facilitate this process, participants placed black dots into the cloud, which were then automatically connected by a black line based on a cubic Bezier smoothing curve. They were asked to place the first dot on the left boundary and the final dot on the right boundary of the graph. In between, participants were allowed to place as many dots as they liked (from left to right) and could remove previously placed dots. There were 50 trials in total. We assessed the average root mean squared distance between participants' predictions (the line they drew) and the mean predictions of each kernel given the data participants had seen, for both interpolation and extrapolation areas. Results are shown in Figure 7.

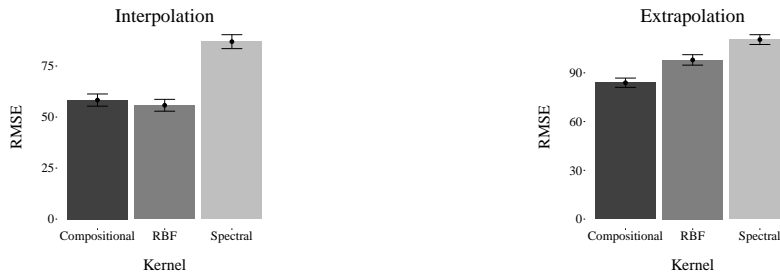

(a) Distance for interpolation drawings.   (b) Distance for extrapolation drawings.

Figure 7: Root mean squared distances. Error bars represent the standard error of the mean.

The mean distance from participants' drawings was significantly higher for the spectral mixture kernel than for the compositional kernel in both interpolation (86.96 vs. 58.33, $t(1291.1) = -6.3$, $p < .001$) and extrapolation areas (110.45 vs 83.91, $t(1475.7) = 6.39$, $p < 0.001$). The radial basis kernel produced similar distances as the compositional kernel in interpolation (55.8), but predicted participants' drawings significantly worse in extrapolation areas (97.9, $t(1459.9) = 3.26$, $p < 0.01$).

## 7    Experiment 4: Assessing predictability

Compositional patterns might also affect the way in which participants perceive functions *a priori* [20]. To assess this, we asked participants to judge how well they thought they could predict 40 different functions that were similar on many measures such as their spectral entropy and their average wavelet distance to each other, but 20 of which were sampled from a compositional and 20 from a spectral mixture kernel. Figure 8 shows a screenshot of the experiment.

50 participants with a mean age of 32 (SD = 7.82) were recruited via Amazon Mechanical Turk and received \$0.5 for their participation. Participants were asked to rate the predictability of different functions. On each trial participants were shown a total of $n_j \in \{50, 60, \ldots, 100\}$ randomly sampled input-output points of a given function and asked to judge how well they thought they could predict the output for a randomly sampled input point on a scale of 0 (not at all) to 100 (very well). Afterwards, they had to rate which of two functions was easier to predict (Figure 8) on a scale from -100 (left graph is definitely easier to predict) to 100 (right graph is definitely easier predict).

As shown in Figure 9, compositional functions were perceived as more predictable than spectral functions in isolation ($t(948) = 11.422$, $p < 0.01$) and in paired comparisons ($t(499) = 13.502$, $p < 0.01$). Perceived predictability increases with the number of observed outputs ($r = 0.23$,

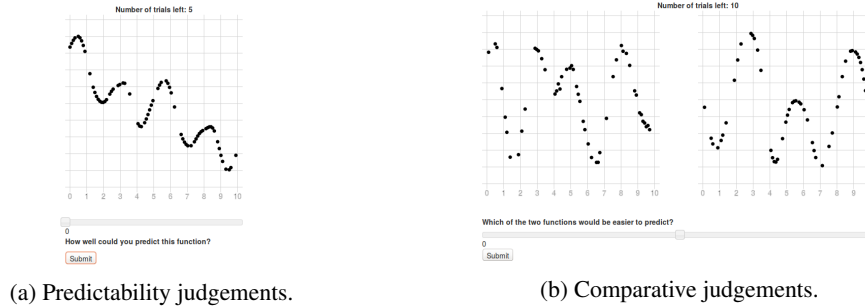

(a) Predictability judgements.                    (b) Comparative judgements.

Figure 8: Screenshot of the predictablity experiment.

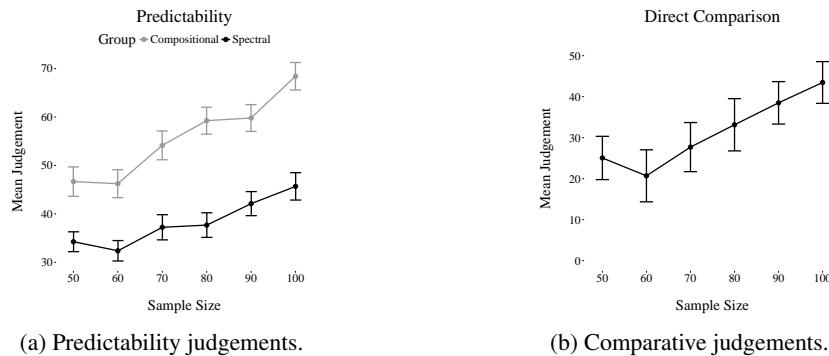

(a) Predictability judgements.                    (b) Comparative judgements.

Figure 9: Results of the predictablity experiment. Error bars represent the standard error of the mean.

$p < 0.01$) and the larger the number of observations, the larger the difference between compositional and spectral mixture functions ($r = 0.14$, $p < 0.01$).

## 8 Discussion

In this paper, we probed human intuitions about functions and found that these intuitions are best described as compositional. We operationalized compositionality using a grammar over kernels within a GP regression framework and found that people prefer extrapolations based on compositional kernels over other alternatives, such as a spectral mixture or the standard radial basis kernel. Two Markov chain Monte Carlo with people experiments revealed that participants converge to extrapolations consistent with the compositional kernels. These findings were replicated when people manually drew the functions underlying observed data. Moreover, participants perceived compositional functions as more predictable than non-compositional – but otherwise similar – ones.

The work presented here is connected to several lines of previous research, most importantly that of Lucas et al. [15], which introduced GP regression as a model of human function learning, and Wilson et al. [22], which attempted to reverse-engineer the human kernel using a spectral mixture. We see our work as complementary; we need both a theory to describe how people make sense of structure as well as a method to indicate what the final structure might look like when represented as a kernel. Our approach also ties together neatly with past attempts to model structure in other cognitive domains such as motion perception [9] and decision making [7].

Our work can be extended in a number of ways. First, it is desirable to more thoroughly explore the space of base kernels and composition operators, since we used an elementary grammar in our analyses that is probably too simple. Second, the compositional approach could be used in traditional function learning paradigms (e.g., [5, 14]) as well as in active input selection paradigms [17]. Another interesting avenue for future research would be to explore the broader implications of compositional function representations. For example, evidence suggests that statistical regularities reduce perceived numerosity [23] and increase memory capacity [2]; these tasks can therefore provide clues about the underlying representations. If compositional functions alter number perception or memory performance to a greater extent than alternative functions, that suggests that our theory extends beyond simple function learning.

## Footnotes

[1] See the following page for an example: `http://learning.eng.cam.ac.uk/carl/mauna`.

# References

[1] L. Bott and E. Heit. Nonmonotonic extrapolation in function learning. *Journal of Experimental Psychology: Learning, Memory, and Cognition*, 30:38–50, 2004.

[2] T. F. Brady, T. Konkle, and G. A. Alvarez. A review of visual memory capacity: Beyond individual items and toward structured representations. *Journal of Vision*, 11:4–4, 2011.

[3] B. Brehmer. Hypotheses about relations between scaled variables in the learning of probabilistic inference tasks. *Organizational Behavior and Human Performance*, 11(1):1–27, 1974.

[4] J. D. Carroll. *Functional learning: The learning of continuous functional mappings relating stimulus and response continua*. Educational Testing Service, 1963.

[5] E. L. DeLosh, J. R. Busemeyer, and M. A. McDaniel. Extrapolation: The sine qua non for abstraction in function learning. *Journal of Experimental Psychology: Learning, Memory, and Cognition*, 23(4):968, 1997.

[6] D. Duvenaud, J. R. Lloyd, R. Grosse, J. B. Tenenbaum, and Z. Ghahramani. Structure discovery in nonparametric regression through compositional kernel search. *Proceedings of the 30th International Conference on Machine Learning*, pages 1166–1174, 2013.

[7] S. J. Gershman, J. Malmaud, J. B. Tenenbaum, and S. Gershman. Structured representations of utility in combinatorial domains. *Decision*, 2016.

[8] S. J. Gershman and Y. Niv. Learning latent structure: carving nature at its joints. *Current Opinion in Neurobiology*, 20:251–256, 2010.

[9] S. J. Gershman, J. B. Tenenbaum, and F. Jäkel. Discovering hierarchical motion structure. *Vision Research*, 2016.

[10] T. L. Griffiths, C. Lucas, J. Williams, and M. L. Kalish. Modeling human function learning with gaussian processes. In *Advances in Neural Information Processing Systems*, pages 553–560, 2009.

[11] R. Grosse, R. R. Salakhutdinov, W. T. Freeman, and J. B. Tenenbaum. Exploiting compositionality to explore a large space of model structures. *Uncertainty in Artificial Intelligence*, 2012.

[12] M. L. Kalish, T. L. Griffiths, and S. Lewandowsky. Iterated learning: Intergenerational knowledge transmission reveals inductive biases. *Psychonomic Bulletin & Review*, 14:288–294, 2007.

[13] C. Kemp and J. B. Tenenbaum. Structured statistical models of inductive reasoning. *Psychological Review*, 116:20–58, 2009.

[14] K. Koh and D. E. Meyer. Function learning: Induction of continuous stimulus-response relations. *Journal of Experimental Psychology: Learning, Memory, and Cognition*, 17:811–836, 1991.

[15] C. G. Lucas, T. L. Griffiths, J. J. Williams, and M. L. Kalish. A rational model of function learning. *Psychonomic bulletin & review*, 22(5):1193–1215, 2015.

[16] M. A. Mcdaniel and J. R. Busemeyer. The conceptual basis of function learning and extrapolation: Comparison of rule-based and associative-based models. *Psychonomic Bulletin & Review*, 12:24–42, 2005.

[17] P. Parpart, E. Schulz, M. Speekenbrink, and B. C. Love. Active learning as a means to distinguish among prominent decision strategies. In *Proceedings of the 37th Annual Meeting of the Cognitive Science Society*, pages 1829–1834, 2015.

[18] C. Rasmussen and C. Williams. *Gaussian Processes for Machine Learning*. MIT Press, 2006.

[19] A. N. Sanborn, T. L. Griffiths, and R. M. Shiffrin. Uncovering mental representations with Markov chain Monte Carlo. *Cognitive Psychology*, 60(2):63–106, 2010.

[20] E. Schulz, J. B. Tenenbaum, D. N. Reshef, M. Speekenbrink, and S. J. Gershman. Assessing the perceived predictability of functions. In *Proceedings of the 37th Annual Meeting of the Cognitive Science Society*, pages 2116–2121. Cognitive Science Society, 2015.

[21] A. G. Wilson and R. P. Adams. Gaussian process kernels for pattern discovery and extrapolation. *arXiv preprint arXiv:1302.4245*, 2013.

[22] A. G. Wilson, C. Dann, C. Lucas, and E. P. Xing. The human kernel. In *Advances in Neural Information Processing Systems*, pages 2836–2844, 2015.

[23] J. Zhao and R. Q. Yu. Statistical regularities reduce perceived numerosity. *Cognition*, 146:217–222, 2016.

